# Limits of Spectral Clustering

**Ulrike von Luxburg**   and   **Olivier Bousquet**
Max Planck Institute for Biological Cybernetics
Spemannstr. 38, 72076 Tübingen, Germany
{ulrike.luxburg,olivier.bousquet}@tuebingen.mpg.de

**Mikhail Belkin**
The University of Chicago, Department of Computer Science
1100 E 58th st., Chicago, USA
misha@cs.uchicago.edu

## Abstract

An important aspect of clustering algorithms is whether the partitions constructed on finite samples converge to a useful clustering of the whole data space as the sample size increases. This paper investigates this question for normalized and unnormalized versions of the popular spectral clustering algorithm. Surprisingly, the convergence of unnormalized spectral clustering is more difficult to handle than the normalized case. Even though recently some first results on the convergence of normalized spectral clustering have been obtained, for the unnormalized case we have to develop a completely new approach combining tools from numerical integration, spectral and perturbation theory, and probability. It turns out that while in the normalized case, spectral clustering usually converges to a nice partition of the data space, in the unnormalized case the same only holds under strong additional assumptions which are not always satisfied. We conclude that our analysis gives strong evidence for the superiority of normalized spectral clustering. It also provides a basis for future exploration of other Laplacian-based methods.

## 1   Introduction

Clustering algorithms partition a given data set into several groups based on some notion of similarity between objects. The problem of clustering is inherently difficult and often lacks clear criteria of "goodness". Despite the difficulties in determining the quality of a given partition, it is still possible to study desirable properties of clustering *algorithms* from a theoretical point of view. In this paper we study the consistency of spectral clustering, which is an important property in the general framework of statistical pattern recognition. A clustering algorithm is consistent if it produces a well-defined (and, hopefully, sensible) partition, given sufficiently many data points. The consistency is a basic sanity check, as an algorithm which is not consistent would change the partition indefinitely as we add points to the dataset, and, consequently, no reasonable small-sample performance could be expected at all. Surprisingly, relatively little research into consistency of clustering algorithms has

been done so far, exceptions being only $k$-centers (Pollard, 1981) and linkage algorithms (Hartigan, 1985).

While finite-sample properties of spectral clustering have been studied from a theoretical point of view (Spielman and Teng, 1996; Guattery and Miller, 1998; Kannan et al., 2000; Ng et al., 2001; Meila and Shi, 2001) we focus on the limit behavior for sample size tending to infinity. In this paper we develop a new strategy to prove convergence results for spectral clustering algorithms. Unlike our previous attempts this strategy allows to obtain results for both normalized and unnormalized spectral clustering. As a first result we can recover the main theorem of von Luxburg et al. (2004), which had been proved with different and more restrictive methods, and, in brief, states that usually normalized spectral clustering converges. We also extend that result to the case of multiple eigenvectors. Our second result concerns the case of unnormalized spectral clustering, for which no convergence properties had been known so far. This case is much more difficult to treat than the normalized case, as the limit operators have a more complicated form. We show that unnormalized spectral clustering also converges, but only under strong additional assumptions. Contrary to the normalized case, those assumptions are not always satisfied, as we can show by constructing an example, and in this case there is no hope for convergence. Even worse, on a finite sample it is impossible to verify whether the assumptions hold or not. As a third result we prove statements about the form of the limit clustering. It turns out that in case of convergence, the structure of the clustering constructed on finite samples is conserved in the limit process. From this we can conclude that if convergence takes place, then the limit clustering presents an intuitively appealing partition of the data space.

It is also interesting to note that several recent methods for semi-supervised and transductive learning are based on eigenvectors of similarity graphs (cf. Belkin and Niyogi, 2003; Chapelle et al., 2003; Zhu et al., 2003). Our theoretical framework can also be applied to investigate the consistency of those algorithms with respect to the unlabeled data.

There is an ongoing debate on the advantages of the normalized versus unnormalized graph Laplacians for spectral clustering. It has been found empirically that the normalized version performs as well or better than the unnormalized version (e.g., Van Driessche and Roose, 1995; Weiss, 1999; in the context of semi-supervised learning see also Zhou et al., 2004). We are now able to provide additional evidence to this effect from a theoretical point of view. Normalized spectral clustering is a well-behaved algorithm which always converges to a sensible limit clustering. Unnormalized spectral clustering on the other hand should be treated with care as consistency can only be asserted under strong assumptions which are not always satisfied and, moreover, are difficult to check in practice.

## 2   Graph Laplacians and spectral clustering on finite samples

In the following we denote by $\sigma(T)$ the spectrum of a linear operator, by $C(\mathcal{X})$ the space of continuous functions on $\mathcal{X}$ with infinity norm, and by $\mathrm{rg}(d)$ the range of a function $d \in C(\mathcal{X})$. For given sample points $X_1, ..., X_n$ drawn iid according to an (unknown) distribution $P$ on some data space $\mathcal{X}$ we denote the empirical distribution by $P_n$. For a non-negative, symmetric similarity function $s : \mathcal{X} \times \mathcal{X} \rightarrow \mathbb{R}$ we define the similarity matrix as $K_n := \big(s(X_i, X_j)\big)_{i,j=1,...,n}$, set $d_i := \sum_{j=1}^{n} s(X_i, X_j)$, and define the degree matrix $D_n$ as the diagonal matrix with entries $d_i$. The unnormalized Laplacian matrix is defined as $L_n := D_n - K_n$, and two common ways of normalizing it are $L'_n := D_n^{-1/2} L_n D_n^{-1/2}$ or $L''_n := D_n^{-1} L_n$. In the following we always arrange the eigenvalues of the Laplacian matrices in non-decreasing order $0 = \lambda_1 \leq \lambda_2 ... \leq \lambda_n$ respecting their multiplicities. In its simplest form, unnormalized (resp. normalized) spectral clustering partitions the sample points $X_i$ into two groups according to whether the $i$-th coordinate of the second eigenvector is larger or smaller than a certain threshold $b \in \mathbb{R}$. Often, instead of considering

only the second eigenvector, one uses the first $r$ eigenvectors (for some small number $r$) simultaneously to obtain a partition into several sets. For an overview of different spectral clustering algorithms see for example Weiss (1999).

## 3   Limit results

In this section we want to state and discuss our main results. The *general assumptions* in the following three theorems are that the data space $\mathcal{X}$ is a compact metric space from which the sample points $(X_i)_{i \in \mathbb{N}}$ are drawn independently according to an unknown probability distribution $P$. Moreover we require the similarity function $s : \mathcal{X} \times \mathcal{X} \rightarrow \mathbb{R}$ to be continuous, and in the normalized case to be bounded away from 0, that is $s(x, y) > l > 0$ for all $x, y \in \mathcal{X}$ and some $l \in \mathbb{R}$. By $d \in C(\mathcal{X})$ we will denote the "degree function", and $U'$ and $U$ will denote the "limit operators" of $L'_n$ and $L_n$ for $n \rightarrow \infty$. The exact definitions of these functions and operators, as well as all further mathematical details, definitions, and proofs will be postponed to Section 4.

Let us start with the first question raised in the introduction: does the spectral clustering constructed on a finite sample converge to a partition of the whole data space if the sample size increases? In the normalized case, convergence results have recently been obtained in von Luxburg et al. (2004). However, those methods were specifically designed for the normalized Laplacian and cannot be used in the unnormalized case. Here we state a convergence result for the normalized case in the form how it can be obtained with our new methods. The theorem is formulated for the symmetric normalization $L'_n$, but it holds similarly for the normalization $L''_n$.

**Theorem 1 (Convergence of normalized spectral clustering)** *Under the general assumptions, if the first $r$ eigenvalues of the limit operator $U'$ have multiplicity 1, then the same holds for the first $r$ eigenvalues of $L'_n$ for sufficiently large $n$. In this case, the first $r$ eigenvalues of $L'_n$ converge to the first $r$ eigenvalues of $U'$, and the corresponding eigenvectors converge almost surely. The partitions constructed by normalized spectral clustering from the first $r$ eigenvectors on finite samples converge almost surely to a limit partition of the whole data space.*

Our new result about the convergence in the unnormalized case is the following:

**Theorem 2 (Convergence of unnormalized spectral clustering)** *Under the general assumptions, if the first $r$ eigenvalues of the limit operator $U$ have multiplicity 1 and are not element of $\mathrm{rg}(d)$, then the same holds for the first $r$ eigenvalues of $\frac{1}{n}L_n$ for sufficiently large $n$. In this case, the first $r$ eigenvalues of $\frac{1}{n}L_n$ converge to the first $r$ eigenvalues of $U$, and the the corresponding eigenvectors converge almost surely. The partitions constructed by unnormalized spectral clustering from the first $r$ eigenvectors on finite samples converge almost surely to a limit partition of the whole data space.*

On the first glance, this theorem looks very similar to Theorem 1: if the general assumptions are satisfied and the first eigenvalues are "nice", then unnormalized spectral clustering converges. However, the difference between Theorems 1 and 2 is what it means for an eigenvalue to be "nice". In Theorem 1 we only require the eigenvalues to have multiplicity 1 (and in fact, if the multiplicity is larger than 1 we can still prove convergence of eigenspaces instead of eigenvectors). In Theorem 2, however, the condition $\lambda \notin \mathrm{rg}(d)$ has to be satisfied. In the proof this is needed to ensure that the eigenvalue $\lambda$ is isolated in the spectrum of $U$, which is a fundamental requirement for applying perturbation theory to the convergence of eigenvectors. If this condition is not satisfied, perturbation theory is *in principle* unsuitable to obtain convergence results for eigenvectors. The reason why this condition appears in the unnormalized case but not in the normalized case lies in the

structure of the respective limit operators, which, surprisingly, is more complicated in the unnormalized case than in the normalized one. In the next section we will construct an example where the second eigenvalue indeed lies within $\mathrm{rg}(d)$. This means that there actually exist situations in which Theorem 2 cannot be applied, and hence unnormalized spectral clustering might not converge.

Now we want to turn to the second question raised in the introduction: In case of convergence, is the limit clustering a reasonable clustering of the whole data space? To answer this question we analyze the structure of the limit operators (for simplicity we state this for the unnormalized case only). Assume that we are given a partition $\mathcal{X} = \cup_{i=1}^{k} \mathcal{X}_i$ of the data space into $k$ disjoint sets. If we order the sample points according to their memberships in the sets $\mathcal{X}_i$, then we can write the Laplacian in form of a block matrix $L_n \simeq (L_{ij,n})_{i,j=1,...,k}$ where each sub-matrix $L_{ij,n}$ contains the rows of $L_n$ corresponding to points in set $\mathcal{X}_i$ and the columns corresponding to points in $\mathcal{X}_j$. In a similar way, the limit operator $U$ can be decomposed into a matrix of operators $U_{ij} : \mathcal{C}(\mathcal{X}_j) \to \mathcal{C}(\mathcal{X}_i)$. Now we can show that for all $i, j = 1, ..., k$ the sub-matrices $\frac{1}{n} L_{ij,n}$ converge to the corresponding sub-operators $U_{ij}$ such that their spectra converge in the same way as in Theorems 1 and 2. This is a very strong result as it means that for every given partition of $\mathcal{X}$, the structure of the operators is preserved in the limit process.

**Theorem 3 (Structure of the limit operators)** *Let $\mathcal{X} = \cup_{i=1}^{k} \mathcal{X}_i$ be a partition of the data space. Let $L_{ij,n}$ be the sub-matrices of $L_n$ introduced above, $U_{ij} : \mathcal{C}(\mathcal{X}_j) \to \mathcal{C}(\mathcal{X}_i)$ the restrictions of $U$ corresponding to the sets $\mathcal{X}_i$ and $\mathcal{X}_j$, and $U'_{ij,n}$ and $U'_{ij}$ the analogous quantities for the normalized case. Then under the general assumptions, $\frac{1}{n} L_{ij,n}$ converges compactly to $U_{ij}$ a.s. and $L'_{ij,n}$ converges compactly to $U'_{ij}$ a.s.*

With this result it is then possible to give a first answer on the question how the limit partitions look like. In Meila and Shi (2001) it has been established that normalized spectral clustering tries to find a partition such that a random walk on the sample points tends to stay within each of the partition sets $\mathcal{X}_i$ instead of jumping between them. With the help of Theorem 3, the same can now be said for the normalized limit partition, and this can also be extended to the unnormalized case. The operators $U'$ and $U$ can be interpreted as diffusion operators on the data space. The limit clusterings try to find a partition such that the diffusion tends to stay within the sets $\mathcal{X}_i$ instead of jumping between them. In particular, the limit partition segments the data space into sets such that the similarity within the sets is high and the similarity between the sets is low, which intuitively is what clustering is supposed to do.

## 4   Mathematical details

In this section we want to explain the general constructions and steps that need to be taken to prove Theorems 1, 2, and 3. However, as the proofs are rather technical we only present proof sketches that convey the overall strategy. Detailed proofs can be found in von Luxburg (2004) where all proofs are spelled out in full length. Moreover, we will focus on the proof of Theorem 2 as the other results can be proved similarly.

To be able to define convergence of linear operators, all operators have to act on the same space. As this is not the case for the matrices $L_n$ for different $n$, for each $L_n$ we will construct a related operator $U_n$ on the space $C(\mathcal{X})$ which will be used instead of $L_n$. In Step 2 we show that the interesting eigenvalues and eigenvectors of $\frac{1}{n} L_n$ and $U_n$ are in a one-to-one relationship. Then we will prove that the $U_n$ converge in a strong sense to some limit operator $U$ on $C(\mathcal{X})$ in Step 3. As we can show in Step 4, this convergence implies the convergence of eigenvalues and eigenvectors of $U_n$. Finally, assembling the parts will finish the proof of Theorem 2.

**Step 1: Construction of the operators $U_n$ on $C(\mathcal{X})$.**

We first define the empirical and true degree functions in $C(\mathcal{X})$ as

$$d_n(x) := \int s(x,y)dP_n(y) \qquad \text{and} \qquad d(x) := \int s(x,y)dP(y).$$

Corresponding to the matrices $D_n$ and $K_n$ we introduce the following multiplication and integral operators on $C(\mathcal{X})$:

$$M_{d_n}f(x) := d_n(x)f(x) \qquad \text{and} \qquad M_d f(x) := d(x)f(x)$$

$$S_n f(x) := \int s(x,y)f(y)dP_n(y) \qquad \text{and} \qquad Sf(x) := \int s(x,y)f(y)dP(y).$$

Note that $d_n(X_i) = \frac{1}{n}d_i$, and for $f \in C(\mathcal{X})$ and $v := (f(X_1),...,f(X_n))'$ it holds that $\frac{1}{n}(D_n v)_i = M_{d_n}f(X_i)$ and $\frac{1}{n}(K_n v)_i = S_n f(X_i)$. Hence the function $d_n$ and the operators $M_{d_n}$ and $S_n$ are the counterparts of the discrete degrees $\frac{1}{n}d_i$ and the matrices $\frac{1}{n}D_n$ and $\frac{1}{n}K_n$. The scaling factor $1/n$ comes from the hidden $1/n$-factor in the empirical distribution $P_n$. The natural pointwise limits of $d_n$, $M_{d_n}$, and $S_n$ for $n \to \infty$ are given by $d$, $M_d$, and $S$. The operators corresponding to the unnormalized Laplacians $\frac{1}{n}L_n = \frac{1}{n}(D_n - K_n)$ and its limit operator are

$$U_n f(x) := M_{d_n}f(x) - S_n f(x) \qquad \text{and} \qquad Uf(x) := M_d f(x) - Sf(x).$$

**Step 2: Relations between $\sigma(\frac{1}{n}L_n)$ and $\sigma(U_n)$.**

**Proposition 4 (Spectral properties)** *1. The spectrum of $U_n$ consists of $\mathrm{rg}(d_n)$, plus some isolated eigenvalues with finite multiplicity. The same holds for $U$ and $\mathrm{rg}(d)$.*

*2. If $f \in C(\mathcal{X})$ is an eigenfunction of $U_n$ with arbitrary eigenvalue $\lambda$, then the vector $v \in \mathbb{R}^n$ with $v_i = f(X_i)$ is an eigenvector of the matrix $\frac{1}{n}L_n$ with eigenvalue $\lambda$.*

*3. If $v$ is an eigenvector of the matrix $\frac{1}{n}L_n$ with eigenvalue $\lambda \notin \mathrm{rg}(d_n)$, then the function $f(x) = \frac{1}{n}(\sum_j s(x,X_j)v_j)/(d_n(x) - \lambda)$ is the unique eigenfunction of $U_n$ with eigenvalue $\lambda$ satisfying $f(X_i) = v_i$.*

*Proof.* It is well-known that the (essential) spectrum of a multiplication operator coincides with the range of the multiplier function. Moreover, the spectrum of a sum of a bounded operator with a compact operator contains the essential spectrum of the bounded operator. Additionally, it can only contain some isolated eigenvalues with finite multiplicity (e.g., Theorem IV.5.35 in Kato, 1966). The proofs of the other parts of this proposition can be obtained by elementary shuffling of eigenvalue equations and will be skipped. ☺

**Step 3: Convergence of $U_n$ to $U$.**

*Dealing with the randomness.* Recall that the operators $U_n$ are random operators as they depend on the given sample points $X_1,...,X_n$ via the empirical distribution $P_n$. One important tool to cope with this randomness will be the following proposition:

**Proposition 5 (Glivenko-Cantelli class)** *Let $(\mathcal{X},d)$ be a compact metric space and $s : \mathcal{X} \times \mathcal{X} \to \mathbb{R}$ continuous. Then $\mathcal{F} := \{s(x,\cdot); \ x \in \mathcal{X}\}$ is a Glivenko-Cantelli class, that is $\sup_{x \in \mathcal{X}} |\int s(x,y)dP_n(y) - \int s(x,y)dP(y)| \to 0$ almost surely.*

*Proof.* This proposition follows from Theorem 2.4.1. of v. d. Vaart and Wellner (1996). ☺

Note that one direct consequence of this proposition is that $\|d_n - d\|_\infty \to 0$ a.s.

*Types of convergence.* Let $E$ be an arbitrary Banach space and $B$ its unit ball. A sequence $(S_n)_n$ of linear operators on $E$ is called *collectively compact* if the set $\bigcup_n S_n B$ is relatively compact in $E$ (with respect to the norm topology). A sequence of operators *converges collectively compactly* if it converges pointwise and if there exists some $N \in I\!N$ such that the operators $(S_n - S)_{n>N}$ are collectively compact. A sequence of operators *converges compactly* if it converges pointwise and if for every sequence $(x_n)_n$ in $B$, the sequence $(S-S_n)x_n$ is relatively compact. See Anselone (1971) and Chatelin (1983) for background reading. A sequence $(x_n)_n$ in $E$ *converges up to a change of sign* to $x \in E$ if there exists a sequence $(a_n)_n$ of signs $a_n \in \{-1, +1\}$ such that the sequence $(a_n x_n)_n$ converges to $x$.

**Proposition 6 ($U_n$ converges compactly to $U$ a.s.)** *Let $\mathcal{X}$ be a compact metric space and $s : \mathcal{X} \times \mathcal{X} \to I\!R$ continuous. Then $U_n$ converges to $U$ compactly a.s.*

*Proof.* (a) $S_n$ *converges to $S$ collectively compactly a.s.* With the help of the Glivenko-Cantelli property in Proposition 5 it is easy to see that $S_n$ converges to $S$ pointwise, that is $\|S_n f - S f\|_\infty \to 0$ a.s. for all $f \in C(\mathcal{X})$. As the limit operator $S$ is compact, to prove that $(S_n - S)_n$ are collectively compact a.s. it is enough to prove that $(S_n)_n$ are collectively compact a.s. This can be done by the Arzela-Ascoli theorem.
  (b) $M_{d_n}$ *converges to $M_d$ in operator norm a.s.* This is a direct consequence of the Glivenko-Cantelli properties of Proposition 5.
  (c) $U_n = S_n - M_{d_n}$ *converges to $U = S - M_d$ compactly a.s.* Both operator norm convergence and collectively compact convergence imply compact convergence (cf. Proposition 3.18 of Chatelin, 1983). Moreover, it is easy to see that the sum of two compactly converging operators converges compactly. ☺

**Step 4: Convergence of the eigenfunctions of $U_n$ to those of $U$.**

It is a result of perturbation theory (see the comprehensive treatment in Chatelin, 1983, especially Section 5.1) that compact convergence of operators implies the convergence of eigenvalues and spectral projections in the following way. If $\lambda$ is an isolated eigenvalue in $\sigma(U)$ with finite multiplicity, then there exists a sequence $\lambda_n \in \sigma(U_n)$ of isolated eigenvalues with finite multiplicity such that $\lambda_n \to \lambda$. If the first $r$ eigenvalues of $T$ have multiplicity 1, then the same holds for the first $r$ eigenvalues of $T_n$ for sufficiently large $n$, and the $i$-th eigenvalues of $T_n$ converge to the $i$-th eigenvalue of $T$. The corresponding eigenvectors converge up to a change of sign. If the multiplicity of the eigenvalues is larger than 1 but finite, then the corresponding eigenspaces converge. Note that for eigenvalues which are not isolated in the spectrum, convergence cannot be asserted, and the same holds for the corresponding eigenvectors (e.g., Section IV.3 of Kato, 1966).
In our case, by Proposition 4 we know that the spectrum of $U$ consists of the whole interval $\mathrm{rg}(d)$, plus eventually some isolated eigenvalues. Hence an eigenvalue $\lambda \in \sigma(U)$ is isolated in the spectrum iff $\lambda \notin \mathrm{rg}(d)$ holds, in which case convergence holds as stated above.

**Step 5: Convergence of unnormalized spectral clustering.**

Now we can to put together the different parts. In the first two steps we transferred the problem of the convergence of the eigenvectors of $\frac{1}{n}L_n$ to the convergence of eigenfunctions of $U_n$. In Step 3 we showed that $U_n$ converges compactly to the limit operator $U$, which according to Step 4 implies the convergence of the eigenfunctions of $U_n$. In terms of the eigenvectors of $\frac{1}{n}L_n$ this means the following: if $\lambda$ denotes the $j$-th eigenvalue of $U$ with eigenfunction $f \in C(\mathcal{X})$ and $\lambda_n$ the $j$-th eigenvalue of $\frac{1}{n}L_n$ with eigenvector $v_n = (v_{n,1}, ..., v_{n,n})'$, then there exists a sequence of signs $a_i \in \{-1, +1\}$ such that $\sup_{i=1,...,n} |a_i v_{n,i} - f(X_i)| \to 0$ a.s. As spectral clustering is constructed from the coordinates of the eigenvectors, this leads to the convergence of spectral clustering in the unnormalized case. This completes the proof of Theorem 2. ☺

The proof for Theorem 1 can be obtained in a very similar way. Here the limit operator is

$$U'f(x) := (I - S')f(x) := f(x) - \int (s(x,y)/\sqrt{d(x)d(y)})f(y)dP(y).$$

The main difference to the unnormalized case is that the operator $M_d$ in $U$ gets replaced by the identity operator $I$ in $U'$. This simplifies matters as one can easily express the spectrum of $(I - S')$ via the spectrum of the compact operator $S'$. From a different point of view, consider the identity operator as the operator of multiplication by the constant one function $\mathbf{1}$. Its range is the single point $\mathrm{rg}(\mathbf{1}) = \{1\}$, and hence the critical interval $\mathrm{rg}(d) \subset \sigma(U)$ shrinks to the point $1 \in \sigma(U')$, which in general is a non-isolated eigenvalue with infinite multiplicity.

Finally, note that it is also possible to prove more general versions of Theorems 1 and 2 where the eigenvalues have finite multiplicity larger than 1. Instead of the convergence of the eigenvectors we then obtain the convergence of the projections on the eigenspaces.

The proof of Theorem 3 works as the ones of the other two theorems. The exact definitions of the operators considered in this case are

$$U'_{ij} : \mathcal{C}(\mathcal{X}_j) \to \mathcal{C}(\mathcal{X}_i), \ \delta_{ij}f_i(x) - \int (s_{ij}(x,y)/\sqrt{d_i(x)d_j(y)})f_j(y)dP_j(y)$$

$$U_{ij} : \mathcal{C}(\mathcal{X}_j) \to \mathcal{C}(\mathcal{X}_i), \ U_{ij}f(x) = \delta_{ij}d_i(x)f_i(x) - \int s_{ij}(x,y)f_j(y)dP_j(y)$$

where $d_i, f_i, P_i$, and $s_{ij}$ denote the restrictions of the functions to $\mathcal{X}_i$ and $\mathcal{X}_i \times \mathcal{X}_j$, respectively, and $\delta_{ij}$ is 1 if $i = j$ and 0 otherwise. For the diffusion interpretation, note that if there exists an ideal partition of the data space (that is, $s(x_i, x_j) = 0$ for $x_i, x_j$ in different sets $\mathcal{X}_i$ and $\mathcal{X}_j$), then the off-diagonal operators $U'_{ij}$ and $U_{ij}$ with $i \neq j$ vanish, and the first $k$ eigenvectors of $U'$ and $U$ can be reconstructed by the piecewise constant eigenvectors of the diagonal operators $U'_{ii}$ and $U_{ii}$. In this situation, spectral clustering recovers the ideal clustering. If there exists no ideal clustering, but there exists a partition such that the off-diagonal operators are "small" and the diagonal operators are "large", then it can be seen by perturbation theory arguments that spectral clustering will find such a partition. The off-diagonal operators can be interpreted as diffusion operators between different clusters (note that even in the unnormalized case, the multiplication operator only appears in the diagonal operators). Hence, constructing a clustering with small off-diagonal operators corresponds to a partition such that few diffusion between the clusters takes place.

Finally, we want to construct an example where the second eigenvalue of $U$ satisfies $\lambda \in \mathrm{rg}(d)$. Let $\mathcal{X} = [1,2] \subset \mathbb{R}$, $s(x,y) := xy$, and $p$ a piecewise constant probability density on $\mathcal{X}$ with $p(x) = c$ if $4/3 \leq x < 5/3$ and $p(x) = (3-c)/2$ otherwise, for some fixed constant $c \in [0,3]$ (e.g., for small $c$ this density has two clearly separated high density regions). The degree function in this case is $d(x) = 1.5x$ (independently of $c$) and has range $[1.5, 3]$ on $\mathcal{X}$. We can see that an eigenfunction of $U$ for eigenvalue $\lambda \notin \mathrm{rg}(d)$ has the form $f(x) = \beta x/(3x - \lambda)$, where the equation $\beta = \int x^2/(3x - \lambda)p(x)dx$ has to be satisfied. This means that $\lambda \notin \mathrm{rg}(d)$ is an eigenvalue of $U$ iff the equation $g(\lambda) := \int_1^2 x^2/(3x - \lambda)p(x)dx \overset{!}{=} 1$ is satisfied. For our simple density function $p$, this integral can be solved analytically. It can then been seen that $g(\lambda) = 1$ is only satisfied for $\lambda = 0$, hence the only eigenvalue outside of $\mathrm{rg}(d)$ is the trivial eigenvalue 0.

Note that in applications of spectral clustering, we do not know the limit operator $U$ and hence cannot test whether its relevant eigenvalues are in its essential spectrum $\mathrm{rg}(d)$ or not. If, for some special reason, one really wants to use unnormalized spectral clustering, one should at least estimate the critical region $\mathrm{rg}(d)$ by $[\min_i d_i/n, \max_i d_i/n]$ and check whether the relevant eigenvalues of $\frac{1}{n}L_n$ are inside or close to this interval or not. This observation then gives an indication whether the results obtained can considered to be reliable or not. However, this observation is not a valid statistical test.

# 5 Conclusions

We have shown that under standard assumptions, normalized spectral clustering always converges to a limit partition of the whole data space which depends only on the probability distribution $P$ and the similarity function $s$. For unnormalized spectral clustering, this can only be guaranteed under the strong additional assumption that the first eigenvalues of the Laplacian do not fall inside the range of the degree function. As shown by our example, this condition has to be taken seriously.

Consistency results are a basic sanity check for behavior of statistical learning algorithms. Algorithms which do not converge cannot be expected to exhibit reliable results on finite samples. Therefore, in the light of our theoretical analysis we assert that the normalized version of spectral clustering should be preferred in practice. This suggestion also extends to other applications of graph Laplacians including semi-supervised learning.

## References

P. Anselone. *Collectively compact operator approximation theory*. Prentice-Hall, 1971.

M. Belkin and P. Niyogi. Using manifold structure for partially labeled classification. In *Advances in Neural Information Processing Systems 15*, 2003.

O. Chapelle, J. Weston, and B. Schölkopf. Cluster kernels for semi-supervised learning. In *Advances in Neural Information Processing Systems 15*, 2003.

F. Chatelin. *Spectral Approximation of Linear Operators*. Academic Press, 1983.

S. Guattery and G. L. Miller. On the quality of spectral separators. *SIAM Journal of Matrix Anal. Appl.*, 19(3), 1998.

J. Hartigan. Statistical theory in clustering. *Journal of classification*, 2:63–76, 1985.

R. Kannan, S. Vempala, and A. Vetta. On clusterings - good, bad and spectral. Technical report, Computer Science Department, Yale University, 2000.

T. Kato. *Perturbation theory for linear operators*. Springer, Berlin, 1966.

M. Meila and J. Shi. A random walks view of spectral segmentation. In *8th International Workshop on Artificial Intelligence and Statistics*, 2001.

A. Ng, M. Jordan, and Y. Weiss. On spectral clustering: Analysis and an algorithm. In *Advances in Neural Information Processing Systems 14*, 2001.

D. Pollard. Strong consistency of k-means clustering. *Ann. of Stat.*, 9(1):135–140, 1981.

D. Spielman and S. Teng. Spectral partitioning works: planar graphs and finite element meshes. In *37th Annual Symposium on Foundations of Computer Science*, 1996.

A. v. d. Vaart and J. Wellner. *Weak Convergence and Empirical Processes*. Springer, 1996.

R. Van Driessche and D. Roose. An improved spectral bisection algorithm and its application to dynamic load balancing. *Parallel Comput.*, 21(1), 1995.

U. von Luxburg. *Statistical Learning with Similarity and Dissimilarity Functions*. PhD thesis, draft, available at http://www.kyb.tuebingen.mpg.de/~ule, 2004.

U. von Luxburg, O. Bousquet, and M. Belkin. On the convergence of spectral clustering on random samples: the normalized case. In *COLT*, 2004.

Y. Weiss. Segmentation using eigenvectors: A unifying view. In *Proceedings of the International Conference on Computer Vision*, pages 975–982, 1999.

D. Zhou, O. Bousquet, T. Lal, J. Weston, and B. Schölkopf. Learning with local and global consistency. In *Advances in Neural Information Processing Systems 16*, 2004.

X. Zhu, Z. Ghahramani, and J. Lafferty. Semi-supervised learning using Gaussian fields and harmonic functions. In *ICML*, 2003.
